# Multiscale Random Fields with Application to Contour Grouping

**Longin Jan Latecki**
Dept. of Computer and Info. Sciences
Temple University, Philadelphia, USA
`latecki@temple.edu`

**ChengEn Lu**
Dept. of Electronics and Info. Eng.
Huazhong Univ. of Sci. and Tech., China
`luchengen@gmail.com`

**Marc Sobel**
Statistics Dept.
Temple University, Philadelphia, USA
`marc.sobel@temple.edu`

**Xiang Bai**
Dept. of Electronics and Info. Eng.
Huazhong Univ. of Sci. and Tech., China
`xiang.bai@gmail.com`

## Abstract

We introduce a new interpretation of multiscale random fields (MSRFs) that admits efficient optimization in the framework of regular (single level) random fields (RFs). It is based on a new operator, called append, that combines sets of random variables (RVs) to single RVs. We assume that a MSRF can be decomposed into disjoint trees that link RVs at different pyramid levels. The append operator is then applied to map RVs in each tree structure to a single RV. We demonstrate the usefulness of the proposed approach on a challenging task involving grouping contours of target shapes in images. It provides a natural representation of multiscale contour models, which is needed in order to cope with unstable contour decompositions. The append operator allows us to find optimal image segment labels using the classical framework of relaxation labeling. Alternative methods like Markov Chain Monte Carlo (MCMC) could also be used.

## 1 Introduction

Random Fields (RFs) have played an increasingly important role in the fields of image denoising, texture discrimination, image segmentation and many other important problems in computer vision. The images analyzed for these purposes typically have significant fractal properties which preclude the use of models operating at a single resolution level. Such models, which aim to minimize mean-squared estimation error, use only second-order image statistics which fail to accurately characterize the images of interest. Multiscale random fields (MSRFs) resolve this problem by using information at many different resolution levels [2, 15, 5]. In [6], a probabilistic model of multiscale conditional random fields (mCRF) was proposed to segment images by labeling pixels using a predefined set of class labels.

The main difference between the proposed interpretation of MSRFs or mCFF as known in the literature, e.g., [2, 15, 6, 5], and the proposed MSRF is the interpretation of the connections between different scales (levels). In the proposed approach, the random variables (RVs) linked by a tree sub-structure across different levels compete for their label assignments, while in the existing approaches the goal is to cooperate in the label assigns, which is usually achieved by averaging. In other words, usually the label assignment of a parent node is enforced to be compatible with the label assignment of its children by averaging. In contrast, in the proposed approach the parent node and all its children compete for the best possible label assignment.

Contour grouping is one of key approaches to object detection and recognition, which is a fundamental goal of computer vision. We introduce a novel MSRF interpretation, and show its benefits in solving the contour grouping problem. The MSRF allows us to cast contour grouping as contour matching. Detection and grouping by shape has been investigated in earlier work. The basic

idea common to all methods is to define distance measures between shapes, and then accurately label and/or classify shapes using these measures. Classical methods, of this type, such as shape contexts [1] and chamfer matching [13] can not cope well with clutter and shape deformations. Some researchers described the shape of the entire object using deformable contour fragments and their relative positions [10, 12], but their detection results are always grassy contour edges. The deformable template matching techniques often require either good initial positions or clean images (or both) to avoid (false) local minima [14, 9]. Recently, Ferrari et al. [4] have used the sophisticated edge detection methods of [8]; the resulting edges are linked to a network of connected contour segments by closing small gaps. Wu et al. [16] proposed an active basis model that provides deformable template consisting of a small number of Gabor wavelet elements allowed to slightly perturb their locations and orientations.

Our grouping is also based on the edge detection of [8], but we do not perform edge linking directly for purposes of grouping. We perform matching a given contour model to edge segments in images. This allows us to perform grouping and detection at the same time. Our method differs from former sampled-points-based matching methods [14, 3]; we match the contour segments from the given contour to segments in edge images directly.

We decompose a given closed contour of a model shape into a group of contour segments, and match the resulting contour segments to edge segments in a given image. Our model contour decomposition is flexible and admits a hierarchical structure, e.g., a parent contour segment is decomposed into two or more child segments. In this way, our model can adapt to different configurations of contour parts in edge images. The proposed MSRF interpretation allows us to formulate the problem of contour grouping as a soft label assignment problem. Since in our approach a parent node and all its children compete for the best possible label assignment, allowing us to examine multiple composite hypotheses of model segments in the image, a successful contour grouping of edge segments is possible even if significant contour parts are missing or are distorted. The competition is made possible by the proposed append operator. It appends the random variables (RVs) representing the parent and all its children nodes to a single new RV. Since the connectivity relation between each pair of model segments is known, the soft label assignment and the competition for best labels make accurate grouping results in real images possible.

We also want to stress that our grouping approach is based on matching of contour segments. The advantages of segment matching over alternative techniques based on point matching are at least twofold: 1) it permits deformable matching (i.e., the global shape will not be changed even when some segments shift or rotate a little); 2) it is more stable than point matching, since contour segments are more informative than points as shape cues.

## 2 Multiscale Random Fields

Given a set of data points $X = \{x_1, \ldots, x_n\}$, the goal of random fields is to find a label assignment $f$ that maximizes the posterior probability $p(f|X)$ (of that assignment):

$$\widehat{f} = \operatorname{argmax}_f p(f|X) \tag{1}$$

Thus, we want to select the label assignment with the largest possible probability given the observed data. Although the proposed method is quite general, for clarity of presentation, we focus on an application of interest to us: contour grouping based on contour part correspondence.

We take the contour of an example shape to be our shape model $S$. We assume that the model is composed of several contour segments $s_1, \ldots, s_m$. In our application, the data points $X = \{x_1, \ldots, x_n\}$ are contour segments extracted by some low level process in a given image. The random field is defined by a sequence of random variables $F = (F_1, \ldots, F_m)$ associated with nodes $s_i$ of the model graph $F$ represents the mapping of the nodes (model segments) $S = \{s_1, , s_m\}$ to the data points $X = \{x_1, \ldots, x_n\}$ (i.e., $F : S \rightarrow X$). We write $F_i = x_j$ to denote the event that the model segment $s_i$ is assigned the image segment $x_j$ by the map F. (Observe that usually the assignment is defined in the reverse direction, i.e., from an image to the model.)

Our goal is to find a label assignment $f = (f_1, \ldots, f_m) \in X^m$ that maximizes the probability $p(f|X) = p(F_1 = f_1, \ldots, F_m = f_m|X)$, i.e.,

$$\widehat{f} = (\widehat{f}_1, \ldots, \widehat{f}_m) = \operatorname*{argmax}_{(f_1,\ldots,f_m)} p(F_1 = f_1, \ldots, F_m = f_m|X) \tag{2}$$

However, the object contour in the given image (which is composed of some subset of segments in $X = \{x_1, \ldots, x_n\}$ may have a different decomposition into contour segments than is the case for the model $s_1, \ldots, s_m$. This is the case, for example, if some parts of the true contour are missing, i.e., some $s_i$ may not correspond to parts in $X$. Therefore, a shape model is needed that can provide robust detection and recognition under these conditions. We introduce such a model by imposing a multiscale structure on contour segments of the model shape. Let the lowest level zero represents the finest subdivision of a given model contour $S$ into the segments $S^0 = \{s_1^0, \ldots, s_{m_0}^0\}$. The $\alpha$ level partition subdivides the contour into the segments $S^\alpha = \{s_1^\alpha, \ldots, s_{m_\alpha}^\alpha\}$ for $\alpha = 1, \ldots, \beta$, where $\beta$ denotes the highest (i.e., most coarse) pyramid level. For each pyramid level $\alpha$, the segments, $S^\alpha$, partition the model contour $S$, i.e., $S = s_1^\alpha \cup \cdots \cup s_{m_\alpha}^\alpha$. The segments $S_\alpha$ in level $\alpha$ refine the segments $S_{\alpha+1}$ in level $\alpha+1$, i.e., segments in the level $\alpha+1$ are unions of one or more consecutive segments in the level $\alpha$. On each level $\alpha$ we have a graph structure $G^\alpha = (S^\alpha, E^\alpha)$, where $E^\alpha$ is the set of edges governing the relations between segments in $S^\alpha$, and we have a forest composed of trees that link nodes at different levels. The number of the trees corresponds to the number of nodes on the highest level $s_1^\beta, \ldots, s_{m_\beta}^\beta$, since each of these nodes is the root of one tree. We denote these trees with $T_1, \ldots, T_{m_\beta}$. For example, in Fig. 1 we have eight segments on the level zero $s_1^0, \ldots, s_8^0$, and four segments on the level one

$$s_1^1 = s_1^0 \cup s_2^0, \quad s_2^1 = s_3^0 \cup s_4^0, \quad s_3^1 = s_5^0 \cup s_6^0, \quad s_4^1 = s_7^0 \cup s_8^0.$$

This construction leads to a tree structure relation among segments at different levels. For example, $T_1$ is a tree with $s_1^1$ (segment 1) as a parent node and with two children $s_1^0, s_2^0$ (segments 5 and 6).

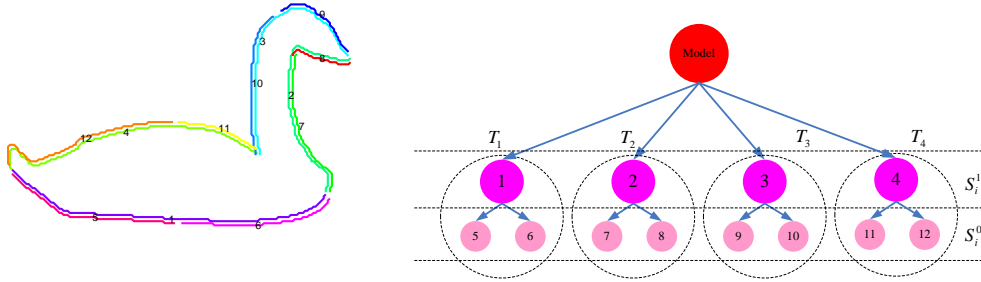

Figure 1: An example of a multiscale random field structure.

We associate a random variable $F_i^\alpha$ with each segment $s_i^\alpha$. The range of each random variable $F_i^\alpha$ is the set of contour segments $X = \{x_1, \ldots, x_n\}$ extracted in a given image. The random variables inherit the tree structure from the corresponding model segments. Thus, we obtain a multiscale random field with random variables (RVs)

$$F = (F_1^0, \ldots, F_{m_0}^0, \ldots, F_1^\alpha, \ldots, F_{m_\alpha}^\alpha, \ldots, F_1^\beta, \ldots, F_{m_\beta}^\beta), \qquad (3)$$

the relational structure (RS) $G^\alpha = (S^\alpha, E^\alpha)$, and trees $T_1, \ldots, T_{m_\beta}$. Our goal remains the same as stated in (2), but the graph structure of the underlying RF is significantly more complicated by the introduction of the multiscale tree relations. Therefore, the maximization in (2) is significantly more complicated as well. Usually, the computation in multiscale random fields is based on modeling the dependencies between the random variables related by the (aforementioned) tree structures.

In the proposed approach, we do not explicitly model these tree structure dependencies. Instead, we build relations between them using the construction of a new random variable that explicitly relates all random variables in each given tree. We introduce a new operator acting on random variables, called **append** operator. The operator combines a given set of random variables $\mathcal{Y} = \{Y_1, \ldots, Y_k\}$ into a single random variable denoted

$$\oplus \mathcal{Y} = Y_1 \oplus \cdots \oplus Y_k. \qquad (4)$$

For simplicity, we assume, in the definition below, that $\{Y_1, \ldots, Y_k\}$ are discrete random variables taking values in the set $X = \{x_1, \ldots, x_n\}$. Our definition can be easily generalized to continuous random variables. The append random variable, $\oplus \mathcal{Y}$, with distribution defined below, takes values in the set of pairs, $\{1, \ldots, k\} \times X$. The distribution of the random variable $\oplus \mathcal{Y}$ is given by,

$$p(\oplus \mathcal{Y} = (i, x_j)) = \frac{1}{k} \cdot p(Y_i = x_j), \qquad (5)$$

where index $i$ is over the RVs and index $j$ is over the labels. The intuition behind this construction can be explained by the following simple example. Let $Y_1, Y_2$ be two discrete random variables with distributions

$$(p(Y_1 = 1), p(Y_1 = 2), p(Y_1 = 3)) \text{ and } (p(Y_2 = 1), p(Y_2 = 2), p(Y_2 = 3)), \tag{6}$$

then the distribution of $Y_1 \oplus Y_2$ is simply given by vector

$$1/2 \cdot (p(Y_1 = 1), p(Y_1 = 2), p(Y_1 = 3), p(Y_2 = 1), p(Y_2 = 2), p(Y_2 = 3)). \tag{7}$$

Armed with this construction, we return to our multiscale RF with RVs in (3). Recall that the RVs representing the nodes on the highest level $F_1^\beta, \ldots, F_{m_\beta}^\beta$ are the roots of trees $T_1, \ldots, T_{m_\beta}$. By slightly abusing our notation, we define $\oplus T_i$ as the append of all random variables that are nodes of tree $T_i$. This construction allows us to reduce the multiscale RF with RVs in (3) to a RF with RVs

$$T = (\oplus T_1, \ldots, \oplus T_{m_\beta}). \tag{8}$$

The graph structure of this new RF is defined by graph $G = (T, E)$ such that

$$(\oplus T_i, \oplus T_j) \in E \text{ iff } \exists_\alpha \exists_{a,b} (F_a^\alpha, F_b^\alpha) \in E^\alpha \text{ and } F_a^\alpha \in \oplus T_i \text{ and } F_b^\alpha \in \oplus T_j \tag{9}$$

In simple words, $\oplus T_i$ and $\oplus T_j$ are related in $G$ iff on some level $\alpha$ both trees have related random variables.

The construction in (8) and (9) maps a multiscale RF to a single level RF, i.e., to a random field with a simple graph structure $G$. The intuition is that we collapse all graphs $G^\alpha = (S^\alpha, E^\alpha)$ for $\alpha = 1, \ldots, \beta$ to a single graph $G = (T, E)$ by gluing all RVs in each tree $T_i$ to a single RV $\oplus T_i$. Consequently, any existing RF optimization method can be applied to compute

$$\hat{t} = (\hat{t}_1, \ldots, \hat{t}_{m_\beta}) = \operatorname*{argmax}_{(t_1, \ldots, t_{m_\beta})} p(\oplus T_1 = t_1, \ldots, \oplus T_{m_\beta} = t_{m_\beta} | X). \tag{10}$$

We observe that when optimizing the new RF in (10), we can simply perform separate optimizations on each level, i.e., on each level $\alpha$ we optimize (8) with respect to the graph structure $G_\alpha$. Hence at each level $\alpha$ we choose the maximum aposteriori estimate associated with the random field at that level. Our key contribution is the fact that these optimizing estimators are linked by the internal structure of the RVs $\oplus T_i$.

After optimizing a regular RF in (10) that contains append RVs, we obtain as the solution updated distributions of the append RVs. From them, we can easily reconstruct the updated distributions of the original RVs from the multiscale RF in (2) by the construction of the append RVs. For example, if we obtain $(\frac{1}{10}, \frac{3}{5}, \frac{1}{10}, 0, \frac{1}{10}, \frac{1}{10})$ as the updated distribution of some RV $Y_1 \oplus Y_2$, then we can easily derive the updated distributions of $Y_1, Y_2$ as

$$(p(Y_1 = 1) = \frac{1}{8}, \ p(Y_1 = 2) = \frac{3}{4}, \ p(Y_1 = 3) = \frac{1}{8}) \ \& \ (p(Y_2 = 1) = 0, \ p(Y_2 = 2) = \frac{1}{2}, \ p(Y_2 = 3) = \frac{1}{2})$$

To obtain the distributions of the compound RVs $Y_1, Y_2$, we only need to ensure that both distributions of $Y_1$ and $Y_2$ sum to one. Since we are usually interested in selecting a variable assignment with maximum posterior probability (10), we do not need to derive these distributions. Consequently, in this example, it is sufficient for us to determine that the assignment of $Y_1$ to label 2 maximizes $Y_1 \oplus Y_2$.

Going back to our application in contour grouping, the RV $\oplus T_2$ is an append of three RVs representing segments 2, 7, 8 in Fig. 1. We observe that RVs appended to $\oplus T_2$ compete in the label assignment. For example, if a given assignment of RV $\oplus T_2$ to an image segment, say $x_5$, maximizes $\oplus T_2$, then, by the position in the discrete distribution of $\oplus T_2$, we can clearly identify which RV is the *winner*, i.e., which of the model segments 2, 7, 8 is assigned to image segment $x_5$. We can also make this competition soft (with more then one *winner*) if we select local maxima of the discrete distribution of $\oplus T_2$, which may lead to assigning more than one of model segments 2, 7, 8 to image segments. In the computation model presented in the next section, we focus on finding a global maximum for each RV $\oplus T_i$.

## 3 Computing the label assignment with relaxation labeling

There exist several approaches to compute the assignment $f$ that optimizes the relational structure of a given RF [7], i.e., approaches that solve Eq. (10), which is our formulation of the general RF Eq. (2). In our implementation, we use a particularly simple approach of relaxation labeling introduced by Rosenfeld et al. in [11]. However, a more powerful class of MCMC methods could also be used [7]. In this section, we briefly describe the relaxation labeling (RL) method, and how it fits into our framework.

We recall that our goal is to find a label assignment $t = (t_1, \ldots, t_m)$ that maximizes the probability $p(t|X) = p(\oplus T_1 = t_1, \ldots, \oplus T_m = t_m|X)$ in Eq. (10), where we have shortened $m = m_\beta$. One of the key ideas of using RL is to decompose $p(t|X)$ into individual probabilities $p(\oplus T_a = (i_a, x_j))$, where index $a = 1, \ldots, m$ ranges over the RVs of the RF, index $j = 1, \ldots, n$ ranges over the possible labels, which in our case are the contour segments $X = \{x_1, \ldots, x_n\}$ extracted from a given image, and index $i_a$ ranges over the RVs that are appended to $\oplus T_a$, which we denote with $i_a \in a$. For brevity, we use the notation

$$p_a(i_a, x_j) = p(\oplus T_a = (i_a, x_j)).$$

Going back to our example in Fig. 1, $p_2(7, x_5)$ denotes the probability that contour segment 7 is assigned to an image segment $x_5$, and 2 is the index of RV $\oplus T_2$. We recall that $\oplus T_2$ is an append of three RVs representing segments 2, 7, 8 in Fig. 1. In Section 5, $p_2(7, x_5)$ is modeled as a Gaussian of the shape dissimilarity between model contour segment 7 and image contour segment 5.

As is usually the case for RFs, we also consider binary relations between RVs that are adjacent in the underlying graph structure $G = (T, E)$, which represent conditional probabilities $p(\oplus T_a = (i_a, x_j) \mid \oplus T_b = (i_b, x_k))$. They express the compatibility of these label assignment. Again for brevity, we use notation

$$C_{a,b}((i_a, x_j), (i_b, x_k)) = p(\oplus T_a = (i_a, x_j) \mid \oplus T_b = (i_b, x_k)).$$

For example, $C_{2,3}((7, x_5), (9, x_8))$ models the compatibility of assignment of model segment 7 (part of model tree 2) to image segment $x_5$ with the assignment of model segment 9 (part of model tree 3) to image segment $x_8$. This compatibility is a function of geometric relations between the segments. Since segment 9 is above segment 7 in the model contour, it is reasonable to assign high compatibility only if the same holds for the image segments, i.e., $x_8$ is above $x_5$.

The RL algorithm iteratively estimates the change in the probability $p_a(i_a, x_j)$ by:

$$\delta p_a(i_a, x_j) = \sum_{b=1,\ldots,m:\, b \neq a} \sum_{i_b \in b} \sum_{x_k \in X:\, x_k \neq x_j} C_{a,b}((i_a, x_j), (i_b, x_k)) \cdot p_b(i_b, x_k), \quad (11)$$

where $b$ varies over all append random variables $\oplus T_b$ different form $\oplus T_a$ and $i_b$ varies over all compound RVs that are combined by append to $\oplus T_b$. Then the probability is updated by

$$p_a(i_a, x_j) = \frac{p_a(i_a, x_j)[1 + \delta p_a(i_a, x_j)]}{\sum_{i_a \in a} \sum_{x_k \in X} p_a(i_a, x_k)[1 + \delta p_a(i_a, x_k)]}, \quad (12)$$

The double sum in the denominator simply normalizes the distribution of $\oplus T_a$ so that it sums to one.

The RL algorithm in our framework iterates steps (11) and (12) for all $a = 1, \ldots, m$ (append RVs), all $i_a \in a$, and all labels $x_j \in X$. It can be shown that the RL algorithm is guaranteed to converge, but not necessarily to a global maximum [7].

## 4 A contour grouping example

We provide a simple but real example to illustrate how our multiscale RF framework solves a concrete contour grouping instance. We use the contour model presented in Fig. 1. Let $F_i$ be a RV corresponding to model contour segment $s_i$ for $i = 1, \ldots, 12$. We have two levels $S^0 = \{F_5, \ldots, F_{12}\}$ and $S^1 = \{F_1, \ldots, F_4\}$. Both graph structures $G^0$ and $G^1$ are complete graphs. As described in Section 2, we have MSRF with four trees. The append RVs determined by these trees are:

$$\oplus T_1 = F_1 \oplus F_5 \oplus F_6, \ \oplus T_2 = F_2 \oplus F_7 \oplus F_8, \ \oplus T_3 = F_3 \oplus F_9 \oplus F_{10}, \ \oplus T_4 = F_4 \oplus F_{11} \oplus F_{12}$$

We obtain a regular (single level) RF with the four append RVs, $T = (\oplus T_1, \oplus T_2, \oplus T_3, \oplus T_4)$, and with the graph structure $G = (T, E)$ determined by Eq. (9).

Given an image as in Fig. 2(a), we first compute its edge map shown in Fig. 2(b), and use a low level edge linking to obtain edge segments in Fig. 2(c). The 16 edge segments in Fig. 2(c) form our label set $X = \{x_1, x_2, \ldots x_{16}\}$. Our goal is to find label assignment to RVs $\oplus T_a$ for $a = 1, 2, 3, 4$ with maximum posterior probability (10). However, the label set of each append RV is different, e.g., the label set of $\oplus T_1$ is equal to $\{1, 5, 6\} \times X$, where $\oplus T_1 = (1, x_5)$ denotes the assignment of $F_1 = x_5$ representing mapping model segment 1 to image segment 5. Hence $p_1(i_a, x_j) = p(\oplus T_1 = (i_a, x_j))$ for $i_a = 1, j = 5$ denotes the probability of mapping model segment $i_a = 1$ to image segment $j = 5$.

As described in Section 3, we use relaxation labeling to compute the maximum posterior probability (10). Initially, all probabilities $p_a(i_a, x_j)$ are set based on shape similarity between involved model and image segments. The assignments compatibilities are determined using geometric relations described in Section 5. After 200 iterations, RL finds the best assignment for each RV $\oplus T_a$ as Fig. 2(d) illustrates. They are presented in the format RV: model segment → edge segment:
$\oplus T_1 : 1 \to x_{12}; \oplus T_2 : 5 \to x_{10}; \oplus T_3 : 8 \to x_7; \oplus T_4 : 4 \to x_5$.
Observe that many model segments remained unmatched, since there they do not have any corresponding segments in the image 2(c). This very desirable property results from the label assignment competition within each append RV $\oplus T_a$ for $a = 1, 2, 3, 4$. This fact demonstrates one of the main benefits of the propose approach. We stress that we do not use any penalties for non matching, which are usually used in classical RFs (e.g., nil variables in [7]), but are very hard to set in real applications.

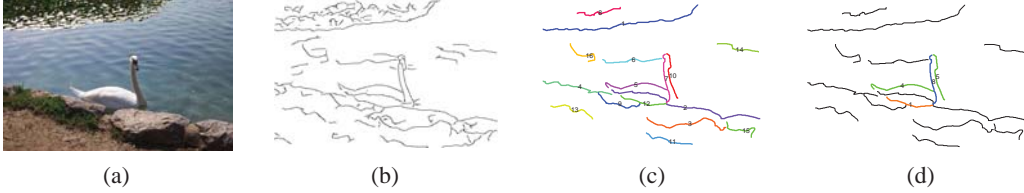

|  (a)  |  (b)  |  (c)  |  (d)  |

Figure 2: (c) The 16 edge segments form our label set $X = \{x_1, x_2, \ldots x_{16}\}$. (d) The numbers and colors indicate the assignment of the model segments from Fig. 1.

## 5 Geometric contour relations

In this section, we provide a brief description of contour segment relations used to assign labels for contour grouping. Two kinds of relations are defined. First, the probability $p_a(i_a, x_j)$ is set to be a Gaussian of shape dissimilarity between model segment $i_a$ and image segment $x_j$. The shape dissimilarity is computed by matching sequences of tangent directions at their sample points. To make our matching scale invariant, we sample each model and image segment with the same number of sample points. We also consider four binary relations to measure the compatibility between a pair of model segments and a pair of image segments: $d^{(1)}(i, i')$ – the maximum distance between the end-points of two contour segments $i$ and $i'$; $d^{(2)}(i, i')$ – the minimum distance between the end-points of two contour segments $i$ and $i'$; $d^{(3)}(i, i')$ – the direction from the mid-point of $i$ to the mid-point of $i'$; $d^{(4)}(i, i')$ – the distance between the mid-points of $i$ and $i'$. To make our relations scale invariant, all distances are normalized by the sum of the lengths of segments $i$ and $i'$. Then the compatibility between pair of model segments $i_a$, $i_b$ and pair of image segments $x_j$, $x_k$ is given by a mixture of Gaussians:

$$C_{a,b}((i_a, x_j), (i_b, x_k)) = \sum_{r=1}^{4} \frac{1}{4} \mathcal{N}(d^{(r)}(i_a, i_b) - d^{(r)}(x_j, x_k), \sigma^{(r)}) \tag{13}$$

## 6 Experimental results

We begin with a comparison between the proposed append MSRF and single level RF. Given an edge map in Fig. 3(b) extracted by edge detector [8], we employ a low level edge linking method to obtain edge segments as shown in 3(c), where the 27 edge segments form our label set $X = \{x_1, \ldots, x_{27}\}$.

Fig. 3(d) illustrates our shape contour model and its two level multiscale structure of 10 contour segments. Fig. 3(e) shows the result of contour grouping obtained in the framework of the proposed

append MSRF. The numbers and colors in indicate the assignment of the model segments. The benefits of the flexible multiscale model structure are clearly visible. Out of 10 model segments, only 4 have corresponding edge segments in the image, and our approach correctly determined a label assignments reflecting this fact.

In contrast, this is not the case for a single level RF. Fig. 3(f) shows a model with a fixed single level structure, and its contour grouping result computed with classical RL can be found in Fig. 3(g). We observe that model segment 2 on giraffe's head has no matching contour in the image, but is nevertheless incorrectly assigned. This wrong assignment influences model contour 4, and leads to another wrong assignment. In the proposed approach, model contours 2 and 3 in Fig. 3(d) compete for label assignments. Since contour 3 finds a good match in the image, we correctly obtain (through our append RV structure) that that there is not match for segment 2.

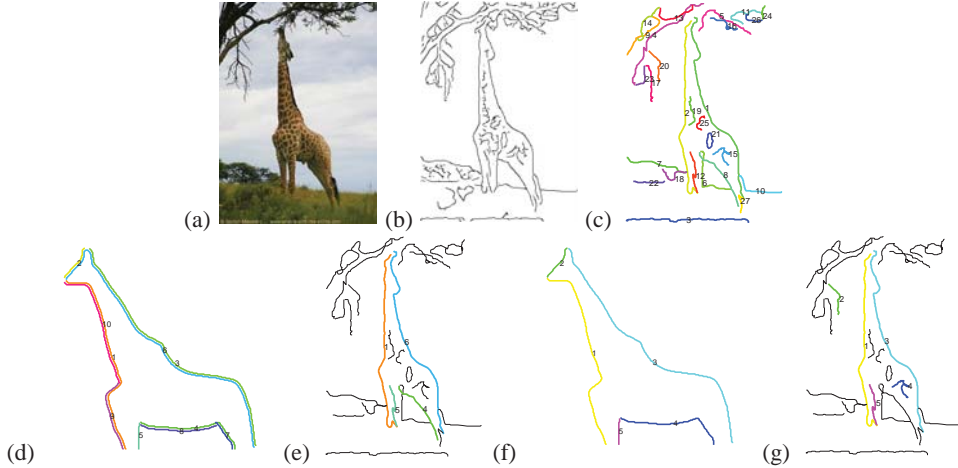

(a)     (b)     (c)

(d)     (e)     (f)     (g)

Figure 3: (d-g) comparison of results obtain by the proposed MSRF to a single level RF.

By mapping the model segments to the image segments, we enforce the existence of a solution. Even if no target shape is present in a given image, our approach will "hallucinate" a matching configuration of edge segments in the image. A standard alternative in the framework of random fields is to use a penalty for non-matching (dummy or null nodes). However, this requires several constants, and it is a highly nontrivial problem to determine their values. In our approach, we can easily distinguish hallucinated contours from true contours, since when the RF optimization is completed, we obtain the assignment of contour segments, i.e., we know a global correspondence between model segments and image segments. Based on this correspondence, we compute global shape similarity, and discard solutions with low global similarity to the model contour. This requires only one threshold on global shape similarity, which is relatively easy to set, and our experimental results verify this fact. In Figs. 4 and 5, we show several examples of contour grouping obtained by the proposed MSRF method on the ETHZ data set [4]. We only use two contour models, the swan model (Fig. 1) and the giraffe model (Fig. 3(d)). Their original images are included as shape models in the ETHZ data set. Model contours are decomposed into segments by introducing break points at high curvature points. Edge contour segments in the test images have been automatically computed by a low level edge linking process. Noise and shape variations cause the edge segments to vary a lot from image to image. We also observe that grouped contours contain internal edge structures.

## 7 Conclusions

Since edges, and consequently, contour parts vary significantly in real images, it is necessary to make decomposition of model contours into segments flexible. The proposed multiscale construction permits us to have a very flexible decomposition that can adapt to different configurations of contour parts in the image. We introduce a novel multiscale random field interpretation based on the append operator that leads to efficient optimization. We applied the new algorithm to the ETHZ data set to illustrate the application potential of the proposed method.

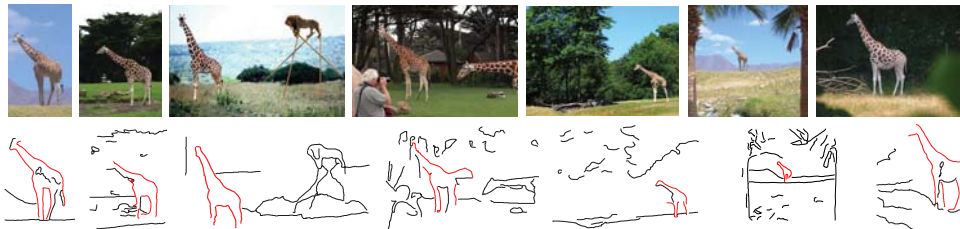

Figure 4: ETHZ data set grouping results for the Giraffe model.

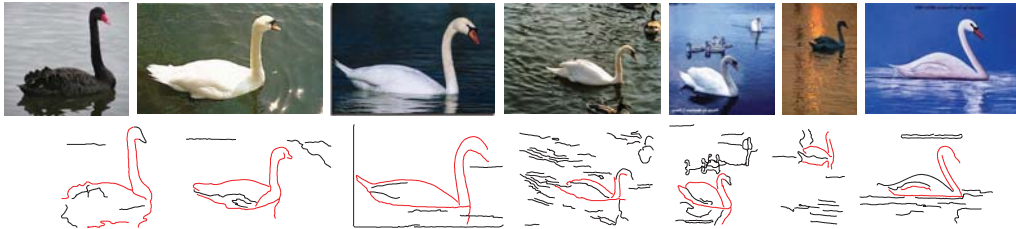

Figure 5: ETHZ data set grouping results for the swan model.

## Acknowledgments

This work was supported in part by the NSF Grants IIS-0534929, IIS-0812118 in the Robust Intelligence Cluster and by the DOE Grant DE-FG52-06NA27508.

## References

[1] S. Belongie, J. Malik, and J. Puzicha. Shape matching and object recognition using shape contexts. *IEEE Trans. Pattern Analysis and Machine Intelligence*, 24:705–522, 2002.

[2] C. A. Bouman and M. Shapiro. A multiscale random field model for bayesian image segmentation. *IEEE Trans. on IP*, 3(2):162–177, 1994.

[3] H. Chui and A. Rangarajan. A new algorithm for non-rigid point matching. In *CVPR*, 2000.

[4] V. Ferrari, L. Fevrier, F. Jurie, and C. Schmid. Groups of adjacent contour segments for object detection. *IEEE Trans. PAMI*, 2008.

[5] A.R. Ferreira and H.K.H.Lee. *Multiscale Modeling: A Bayesian Perspective*. Springer-Verlag, Springer Series in Statistics, 2007.

[6] X. He, R. S. Zemel, and M. A. Carreira-Perpinan. Multiscale conditional random fields for image labeling. In *CVPR*, volume 2, pages 695–702, 2004.

[7] S. Z. Li. *Markov Random Field Modeling in Image Analysis*. Springer-Verlag, Tokyo, 2001.

[8] D. Martin, C. Fowlkes, and J. Malik. Learning to detect natural image boundaries using local birghtness, colour and texture cues. *IEEE Trans. PAMI*, 26:530–549, 2004.

[9] G. McNeill and S. Vijayakumar. Part-based probabilistic point matching using equivalence constraints. In *NIPS*, 2006.

[10] A. Opelt, A. Pinz, and A. Zisserman. A boundary-fragment-model for object detection. In *ECCV*, 2006.

[11] A. Rosenfeld, R. Hummel, and S. Zucker. Scene labeling by relaxation operations. *Trans. on Systems, Man and Cybernetics*, 6:420–433, 1976.

[12] J. Shotton, A. Blake, and R. Cipolla. Contour-based learning for object detection. In *ICCV*, 2005.

[13] A. Thayananthan, B. Stenger, P. H. S. Torr, and R. Cipolla. Shape context and chamfer matching in cluttered scenes. In *CVPR*, 2003.

[14] Z. Tu and A.L. Yuille. Shape matching and recognition using generative models and informative features. In *ECCV*, 2004.

[15] A. S. Willsky. Multiresolution markov models for signal and image processing. *Proceedings of the IEEE*, 90:1396–1458, 2002.

[16] Y. N. Wu, Z. Si, C. Fleming, and S.-C. Zhu. Deformable template as active basis. In *ICCV*, 2007.

